# Online Prediction on Large Diameter Graphs

**Mark Herbster,   Guy Lever,   Massimiliano Pontil**
Department of Computer Science
University College London
Gower Street, London WC1E 6BT, England, UK
{*m.herbster, g.lever, m.pontil*}*@cs.ucl.ac.uk*

## Abstract

We continue our study of online prediction of the labelling of a graph. We show a
fundamental limitation of Laplacian-based algorithms: if the graph has a large di-
ameter then the number of mistakes made by such algorithms may be proportional
to the square root of the number of vertices, even when tackling simple problems.
We overcome this drawback by means of an efficient algorithm which achieves
a logarithmic mistake bound. It is based on the notion of a *spine*, a path graph
which provides a linear embedding of the original graph. In practice, graphs may
exhibit cluster structure; thus in the last part, we present a modified algorithm
which achieves the "best of both worlds": it performs well *locally* in the presence
of cluster structure, and *globally* on large diameter graphs.

## 1   Introduction

We study the problem of predicting the labelling of a graph in the online learning framework. Con-
sider the following game for predicting the labelling of a graph: *Nature* presents a graph; *nature*
queries a vertex $v_{i_1}$; the *learner* predicts $\hat{y}_1 \in \{-1, 1\}$, the label of the vertex; *nature* presents a
label $y_1$; *nature* queries a vertex $v_{i_2}$; the *learner* predicts $\hat{y}_2$; and so forth. The learner's goal is to
minimise the total number of mistakes $M = |\{t : \hat{y}_t \neq y_t\}|$. If nature is adversarial, the learner
will always mispredict, but if nature is regular or simple, there is hope that a learner may make only
a few mispredictions. Thus, a central goal of online learning is to design algorithms whose total
mispredictions can be bounded relative to the complexity of nature's labelling. In [9, 8, 7], the *cut
size* (the number of edges between disagreeing labels) was used as a measure of the complexity of a
graph's labelling, and mistake bounds relative to this and the graph diameter were derived.

The strength of the methods in [8, 7] is in the case when the graph exhibits "cluster structure". The
apparent deficiency of these methods is that they have poor bounds when the graph diameter is large
relative to the number of vertices. We observe that this weakness is not due to insufficiently tight
bounds, but is a problem in their performance. In particular, we discuss an example of a $n$-vertex
labelled graph with a *single edge* between disagreeing label sets. On this graph, sequential prediction
using the common method based upon minimising the Laplacian semi-norm of a labelling, subject to
constraints, incurs $\theta(\sqrt{n})$ mistakes (see Theorem 3). The expectation is that the number of mistakes
incurred by an optimal online algorithm is bounded by $O(\ln n)$.

We solve this problem by observing that there exists an approximate structure-preserving embedding
of any graph into a path graph. In particular the cut-size of any labelling is increased by no more than
a factor of two. We call this embedding a *spine* of the graph. The spine is the foundation on which we
build two algorithms. Firstly we predict directly on the spine with the 1-nearest-neighbor algorithm.
We demonstrate that this equivalent to the Bayes-optimal classifier for a particular Markov random
field. A logarithmic mistake bound for learning on a path graph follows by the Halving algorithm
analysis. Secondly, we use the spine of the graph as a foundation to add a binary *support tree* to the
original graph. This enables us to prove a bound which is the "best of both worlds" – if the predicted
set of vertices has cluster-structure we will obtain a bound appropriate for that case, but if instead,
the predicted set exhibits a large diameter we will obtain a polylogarithmic bound.

**Previous work.** The seminal approach to semi-supervised learning over graphs in [3] is to predict with a labelling which is consistent with a minimum label-separating cut. More recently, the graph Laplacian has emerged as a key object in semi-supervised learning, for example the semi-norm induced by the Laplacian is commonly either directly minimised subject to constraints, or used as a regulariser [14, 2]. In [8, 7] the online graph labelling problem was studied. An aim of those papers was to provide a natural interpretation of the bound on the cumulative mistakes of the kernel perceptron when the kernel is the pseudoinverse of the graph Laplacian – bounds in this case being relative to the cut and (resistance) diameter of the graph. In this paper we necessarily build directly on the very recent results in [7] as those results depend on the resistance diameter of the predicted vertex set as opposed to the whole graph [8]. The online graph labelling problem is also studied in [13], and here the graph structure is not given initially. A slightly weaker logarithmic bound for the online graph labelling problem has also been independently derived via a connection to an online routing problem in the very recent [5].

## 2 Preliminaries

We study the process of predicting a labelling defined on the vertices of a graph. Following the classical *online learning* framework, a sequence of labelled vertices $\{(v_{i_1}, y_1), (v_{i_2}, y_2), \dots\}$, the *trial sequence*, is presented to a learning algorithm such that, on sight of each vertex $v_{i_t}$, the learner makes a prediction $\hat{y}_t$ for the label value, after which the correct label is revealed. This feedback information is then used by the learning algorithm to improve its performance on further examples. We analyse the performance of a learning algorithm in the mistake bound framework [12] – the aim is to minimise the maximum possible cumulative number of mistakes made on the training sequence.

A graph $\mathcal{G} = (V, E)$ is a collection of vertices $V = \{v_1, \dots, v_n\}$ joined by connecting (possibly weighted) edges. Denote $i \sim j$ whenever $v_i$ and $v_j$ are connected so that $E = \{(i,j) : i \sim j\}$ is the set of unordered pairs of connected vertex indices. Associated with each edge $(i,j) \in E$ is a weight $A_{ij}$, so that $A$ is the $n \times n$ symmetric *adjacency matrix*. We say that $\mathcal{G}$ is *unweighted* if $A_{ij} = 1$ for every $(i,j) \in E$ and is 0 otherwise. In this paper, we consider only *connected* graphs – that is, graphs such that there exists a path between any two vertices. The *Laplacian* $G$ of a graph $\mathcal{G}$ is the $n \times n$ matrix $G = D - A$, where $D$ is the diagonal *degree matrix* such that $D_{ii} = \sum_j A_{ij}$. The quadratic form associated with the Laplacian relates to the *cut size* of graph labellings.

**Definition 1.** *Given a labelling $u \in \mathbb{R}^n$ of $\mathcal{G} = (V, E)$ we define the cut size of $u$ by*

$$\Phi_{\mathcal{G}}(u) = \frac{1}{4} u^T G u = \frac{1}{4} \sum_{(i,j) \in E} A_{ij}(u_i - u_j)^2. \tag{1}$$

*In particular, if $u \in \{-1, 1\}^n$ we say that a cut occurs on edge $(i,j)$ if $u_i \neq u_j$ and $\Phi_{\mathcal{G}}(u)$ measures the number of cuts.*

We evaluate the performance of prediction algorithms in terms of the cut size and the *resistance diameter* of the graph. There is an established natural connection between graphs and resistive networks where each edge $(i,j) \in E$ is viewed as a resistor with resistance $1/A_{ij}$ [4]. Thus the *effective resistance* $r_{\mathcal{G}}(v_i, v_j)$ between vertex $v_i$ and $v_j$ is the potential difference needed to induce a unit current flow between $v_i$ and $v_j$. The effective resistance may be computed by the formula [11]

$$r_{\mathcal{G}}(v_i, v_j) = (e_i - e_j)^T G^+ (e_i - e_j), \tag{2}$$

where "$+$" denotes the pseudoinverse and $e_1, \dots, e_n$ are the canonical basis vectors of $\mathbb{R}^n$. The resistance diameter of a graph $R_{\mathcal{G}} := \max_{v_i, v_j \in V} r_{\mathcal{G}}(v_i, v_j)$ is the maximum effective resistance between any pair of vertices on the graph.

## 3 Limitations of online minimum semi-norm interpolation

As we will show, it is possible to develop online algorithms for predicting the labelling of a graph which have a mistake bound that is a logarithmic function of the number of vertices. Conversely, we first highlight a deficiency in a standard Laplacian based method for predicting a graph labelling.

Given a partially labelled graph $\mathcal{G} = (V, E)$ with $|V| = n$ – that is, such that for some $\ell \leq n$, $y_\ell \in \{-1, 1\}^\ell$ is a labelling defined on the $\ell$ vertices $V_\ell = \{v_{i_1}, v_{i_2}, \dots, v_{i_\ell}\}$ – the *minimum semi-norm interpolant* is defined by

$$\bar{y} = \operatorname{argmin}\{u^T G u : u \in \mathbb{R}^n, u_{i_k} = y_k, k = 1, \dots, \ell\}.$$

We then predict using $\hat{y}_i = \mathrm{sgn}(\bar{y}_i)$, for $i = 1, \ldots, n$.

The common justification behind the above learning paradigm [14, 2] is that minimizing the cut (1) encourages neighbouring vertices to be similarly labelled. However, we now demonstrate that in the online setting such a regime will perform poorly on certain graph constructions – there exists a trial sequence on which the method will make at least $\theta(\sqrt{n})$ mistakes.

**Definition 2.** *An octopus graph of size $d$ is defined to be $d$ path graphs (the tentacles) of length $d$ (that is, with $d + 1$ vertices) all adjoined at a common end vertex, to which a further single head vertex is attached, so that $n = |V| = d^2 + 2$. This corresponds to the graph $\mathcal{O}_{1,d,d}$ discussed in [8].*

**Theorem 3.** *Let $\mathcal{G} = (V, E)$ be an octopus graph of size $d$ and $\boldsymbol{y} = (y_1, \ldots, y_{|V|})$ the labelling such that $y_i = 1$ if $v_i$ is the head vertex and $y_i = -1$ otherwise. There exists a trial sequence for which online minimum semi-norm interpolation makes $\theta(\sqrt{|V|})$ mistakes.*

*Proof.* Let the first query vertex be the head vertex, and let the end vertex of a tentacle be queried at each subsequent trial. We show that this strategy forces at least $d$ mistakes. The solution to the minimum semi-norm interpolation with boundary values problem is precisely the *harmonic solution* [4] $\bar{\boldsymbol{y}}$ (that is, for every unlabeled vertex $v_j$, $\sum_{i=1}^{n} A_{ij}(\bar{y}_i - \bar{y}_j) = 0$). If the graph is connected $\bar{\boldsymbol{y}}$ is unique and the graph labelling problem is identical to that of identifying the potential at each vertex of a resistive network defined on the graph where each edge corresponds to a resistor of 1 unit; the harmonic principle corresponds to Kirchoff's current law in this case. Using this analogy, suppose that the end points of $k < d$ tentacles are labelled and that the end vertex $v_q$ of an unlabelled tentacle is queried. Suppose a current of $k\lambda$ flows from the head to the body of the graph. By Kirchoff's law, a current of $\lambda$ flows along each labelled tentacle (in order to obey the harmonic principle at every vertex it is clear that no current flows along the unlabelled tentacles). By Ohm's law $\lambda = \frac{2}{d+k}$. Minimum semi-norm interpolation therefore results in the solution

$$\bar{y}_q = 1 - \frac{2k}{d + k} \geq 0 \text{ iff } k \leq d.$$

Hence the minimum semi-norm solution predicts incorrectly whenever $k < d$ and the algorithm makes at least $d$ mistakes. $\qquad\square$

The above demonstrates a limitation in the method of online Laplacian minimum semi-norm interpolation for predicting a graph labelling – the mistake bound can be proportional to the square root of the number of data points. We solve these problems in the following section.

## 4 A linear graph embedding

We demonstrate a method of embedding data represented as a connected graph $\mathcal{G}$ into a path graph, we call it a *spine* of $\mathcal{G}$, which partially preserves the structure of $\mathcal{G}$. Let $\mathbb{P}_n$ be the set of path graphs with $n$ vertices. We would like to find a path graph with the same vertex set as $\mathcal{G}$, which solves

$$\min_{\mathcal{P} \in \mathbb{P}_n} \max_{\boldsymbol{u} \in \{-1,1\}^n} \frac{\Phi_{\mathcal{P}}(\boldsymbol{u})}{\Phi_{\mathcal{G}}(\boldsymbol{u})}.$$

If a Hamiltonian path $\mathcal{H}$ of $\mathcal{G}$ (a path *on* $\mathcal{G}$ which visits each vertex precisely once) exists, then the approximation ratio is $\frac{\Phi_{\mathcal{H}}(\boldsymbol{u})}{\Phi_{\mathcal{G}}(\boldsymbol{u})} \leq 1$. The problem of finding a Hamiltonian path is NP-complete however, and such a path is not guaranteed to exist. As we shall see, a spine $\mathcal{S}$ of $\mathcal{G}$ may be found efficiently and satisfies $\frac{\Phi_{\mathcal{S}}(\boldsymbol{u})}{\Phi_{\mathcal{G}}(\boldsymbol{u})} \leq 2$.

We now detail the construction of a spine of a graph $\mathcal{G} = (V, E)$, with $|V| = n$. Starting from any node, $\mathcal{G}$ is traversed in the manner of a *depth-first search* (that is, each vertex is fully explored before backtracking to the last unexplored vertex), and an ordered list $V_{\mathcal{L}} = \{v_{l_1}, v_{l_2}, \ldots, v_{l_{2m+1}}\}$ of the vertices ($m \leq |E|$) in the order that they are visited is formed, allowing repetitions when a vertex is visited more than once. Note that each edge in $E_{\mathcal{G}}$ is traversed no more than twice when forming $V_{\mathcal{L}}$. Define an edge multiset $E_{\mathcal{L}} = \{(l_1, l_2), (l_2, l_3), \ldots, (l_{2m}, l_{2m+1})\}$ – the set of pairs of consecutive vertices in $V_{\mathcal{L}}$. Let $\boldsymbol{u}$ be an arbitrary labelling of $\mathcal{G}$ and denote, as usual, $\Phi_{\mathcal{G}}(\boldsymbol{u}) = \frac{1}{4} \sum_{(i,j) \in E_{\mathcal{G}}} (u_i - u_j)^2$ and $\Phi_{\mathcal{L}}(\boldsymbol{u}) = \frac{1}{4} \sum_{(i,j) \in E_{\mathcal{L}}} (u_i - u_j)^2$. Since the multiset $E_{\mathcal{L}}$ contains every element of $E_{\mathcal{G}}$ no more than twice, $\Phi_{\mathcal{L}}(\boldsymbol{u}) \leq 2\Phi_{\mathcal{G}}(\boldsymbol{u})$.

We then take any subsequence $V'_{\mathcal{L}}$ of $V_{\mathcal{L}}$ containing every vertex in $V$ exactly once. A spine $\mathcal{S} = (V, E_{\mathcal{S}})$ is a graph formed by connecting each vertex in $V$ to its immediate neighbours in

the subsequence $V'_{\mathcal{L}}$ with an edge. Since a cut occurs between connected vertices $v_i$ and $v_j$ in $\mathcal{S}$ only if a cut occurs on some edge in $E_{\mathcal{L}}$ located between the corresponding vertices in the list $V_{\mathcal{L}}$ we have

$$\Phi_{\mathcal{S}}(\boldsymbol{u}) \leq \Phi_{\mathcal{L}}(\boldsymbol{u}) \leq 2\Phi_{\mathcal{G}}(\boldsymbol{u}). \tag{3}$$

Thus we have reduced the problem of learning the cut on a generic graph to that of learning the cut on a path graph. In the following we see that 1-nearest neighbour (1-NN) algorithm is a Bayes optimal algorithm for this problem. Note that the 1-NN algorithm does not perform well on general graphs; on the octopus graph discussed above, for example, it can make at least $\theta(\sqrt{n})$ mistakes, and even $\theta(n)$ mistakes on a related graph construction [8].

## 5   Predicting with a spine

We consider implementing the 1-NN algorithm on a path graph and demonstrate that it achieves a mistake bound which is logarithmic in the length of the line. Let $\mathcal{G} = (V, E)$ be a path graph, where $V = \{v_1, v_2, \ldots, v_n\}$ is the set of vertices and $E = \{(1,2),(2,3),\ldots,(n-1,n)\}$. The nearest neighbour algorithm, in the standard online learning framework described above, attempts to predict a graph labelling by producing, for each query vertex $v_{i_t}$, the prediction $\hat{y}_t$ which is consistent with the label of the closest labelled vertex (and predicts randomly in the case of a tie).

**Theorem 4.** *Given the task of predicting the labelling of any unweighted, $n$-vertex path graph $\mathcal{P}$ in the online framework, the number of mistakes, $M$, incurred by the 1-NN algorithm satisfies*

$$M \leq \Phi_{\mathcal{P}}(\boldsymbol{u}) \log_2 \left( \frac{n-1}{\Phi_{\mathcal{P}}(\boldsymbol{u})} \right) + \frac{\Phi_{\mathcal{P}}(\boldsymbol{u})}{\ln 2} + 1, \tag{4}$$

*where $\boldsymbol{u} \in \{-1,1\}^n$ is any labelling consistent with the trial sequence.*

*Proof.* We shall prove the result by noting that the *Halving algorithm* [1] (under certain conditions on the probabilities assigned to each hypothesis) implements the nearest neighbour algorithm on a path graph. Given any input space $X$ and finite binary concept class $\mathcal{C} \subset \{-1,1\}^{|X|}$, the Halving algorithm learns any target concept $c^* \in \mathcal{C}$ as follows. Each *hypothesis* $c \in \mathcal{C}$ is given an associated probability $p(c)$. A sequence of labelled examples $\{(x_1, y_1), \ldots, (x_{t-1}, y_{t-1})\} \subset X \times \{-1,1\}$, is revealed in accordance with the usual online framework. Let $\mathcal{F}_t$ be the set of feasible hypotheses at trial $t$; $\mathcal{F}_t = \{c \,:\, c(x_s) = y_s \, \forall s < t\}$. Given an unlabelled example $x_t \in X$ at trial $t$ the predicted label $\hat{y}_t$ is that which agrees with the *majority vote* – that is, such that $\frac{\sum_{c \in \mathcal{F}_t, c(x_t)=\hat{y}_t} p(c)}{\sum_{c \in \mathcal{F}_t} p(c)} > \frac{1}{2}$ (and it predicts randomly if this is equal to $\frac{1}{2}$). It is well known [1] that the Halving algorithm makes at most $M_H$ mistakes with

$$M_H \leq \log_2 \left( \frac{1}{p(c^*)} \right). \tag{5}$$

We now define a probability distribution over the space of all labellings $\boldsymbol{u} \in \{-1,1\}^n$ of $\mathcal{P}$ such that the Halving algorithm with these probabilities implements the nearest neighbour algorithm. Let a cut occur on any given edge with probability $\alpha$, independently of all other cuts; $\mathrm{Prob}(u_{i+1} \neq u_i) = \alpha$ $\forall i < n$. The position of all cuts fixes the labelling up to flipping every label, and each of these two resulting possible arrangements are equally likely. This recipe associates with each possible labelling $\boldsymbol{u} \in \{-1,1\}^n$ a probability $p(\boldsymbol{u})$ which is a function of the labelling's cut size

$$p(\boldsymbol{u}) = \frac{1}{2}\alpha^{\Phi_{\mathcal{P}}(\boldsymbol{u})}(1-\alpha)^{n-1-\Phi_{\mathcal{P}}(\boldsymbol{u})}. \tag{6}$$

This induces a full joint probability distribution on the space of vertex labels. In fact (6) is a *Gibbs measure* and as such defines a *Markov random field* over the space of vertex labels [10]. The mass function $p$ therefore satisfies the Markov property

$$p(u_i = \gamma \mid u_j = \gamma_j \, \forall j \neq i) = p(u_i = \gamma \mid u_j = \gamma_j \, \forall j \in N_i), \tag{7}$$

where here $N_i$ is the set of vertices neighbouring $v_i$ – those connected to $v_i$ by an edge. We will give an equivalent Markov property which allows a more general conditioning to reduce to that over *boundary vertices*.

**Definition 5.** *Given a path graph $\mathcal{P} = (V, E)$, a set of vertices $V' \subset V$ and a vertex $v_i \in V$, we define the boundary vertices $v_\ell, v_r$ (either of which may be vacuous) to be the two vertices in $V'$ that are closest to $v_i$ in each direction along the path; its nearest neighbours in each direction.*

The distribution induced by (6) satisfies the following Markov property; given a partial labelling of $\mathcal{P}$ defined on a subset $V' \subset V$, the label of any vertex $v_i$ is independent of all labels on $V'$ except those on the vertices $v_\ell, v_r$ (either of which could be vacuous)

$$p(u_i = \gamma \mid u_j = \gamma_j, \; \forall j : v_j \in V') \quad = \quad p(u_i = \gamma \mid u_\ell = \gamma_\ell, \; u_r = \gamma_r). \tag{8}$$

Given the construction of the probability distribution formed by independent cuts on graph edges, we can evaluate conditional probabilities. For example, $p(u_j = \gamma \mid u_k = \gamma)$ is the probability of an even number of cuts between vertex $v_j$ and vertex $v_k$. Since cuts occur with probability $\alpha$ and there are $\binom{|k-j|}{s}$ possible arrangements of $s$ cuts we have

$$p(u_j = \gamma \mid u_k = \gamma) = \sum_{s \text{ even}} \binom{|k-j|}{s} \alpha^s (1-\alpha)^{|k-j|-s} = \frac{1}{2}(1 + (1-2\alpha)^{|k-j|}). \tag{9}$$

Likewise we have that

$$p(u_j \neq \gamma \mid u_k = \gamma) = \sum_{s \text{ odd}} \binom{|k-j|}{s} \alpha^s (1-\alpha)^{|k-j|-s} = \frac{1}{2}(1 - (1-2\alpha)^{|k-j|}). \tag{10}$$

Note also that for any single vertex we have $p(u_i = \gamma) = \frac{1}{2}$ for $\gamma \in \{-1, 1\}$.

**Lemma 6.** *Given the task of predicting the labelling of an $n$-vertex path graph online, the Halving algorithm, with a probability distribution over the labellings defined as in (6) and such that $0 < \alpha < \frac{1}{2}$, implements the nearest neighbour algorithm.*

*Proof.* Suppose that $t - 1$ trials have been performed so that we have a partial labelling of a subset $V' \subset V$, $\{(v_{i_1}, y_1), (v_{i_2}, y_2), \ldots, (v_{i_{t-1}}, y_{t-1})\}$. Suppose the label of vertex $v_{i_t}$ is queried so that the Halving algorithm makes the following prediction $\hat{y}_t$ for vertex $v_{i_t}$: $\hat{y}_t = y$ if $p(u_{i_t} = y \mid u_{i_j} = y_j \; \forall \, 1 \leq j < t) > \frac{1}{2}$, $\hat{y}_t = -y$ if $p(u_{i_t} = y \mid u_{i_j} = y_j \; \forall \, 1 \leq j < t) < \frac{1}{2}$ (and predicts randomly if this probability is equal to $\frac{1}{2}$). We first consider the case where the conditional labelling includes vertices on both sides of $v_{i_t}$. We have, by (8), that

$$
\begin{aligned}
p(u_{i_t} = y \mid u_{i_j} = y_j \; \forall \, 1 \leq j < t) &= p(u_{i_t} = y \mid u_\ell = y_{\tau(\ell)}, u_r = y_{\tau(r)}) \\
&= \frac{p(u_\ell = y_{\tau(\ell)} \mid u_r = y_{\tau(r)}, u_{i_t} = y) p(u_r = y_{\tau(r)}, u_{i_t} = y)}{p(u_\ell = y_{\tau(\ell)}, u_r = y_{\tau(r)})} \\
&= \frac{p(u_\ell = y_{\tau(\ell)} \mid u_{i_t} = y) p(u_r = y_{\tau(r)} \mid u_{i_t} = y)}{p(u_\ell = y_{\tau(\ell)} \mid u_r = y_{\tau(r)})}
\end{aligned} \tag{11}
$$

where $v_\ell$ and $v_r$ are the boundary vertices and $\tau(\ell)$ and $\tau(r)$ are trials at which vertices $v_\ell$ and $v_r$ are queried, respectively. We can evaluate the right hand side of this expression using (9, 10). To show equivalence with the nearest neighbour method whenever $\alpha < \frac{1}{2}$, we have from (9, 10, 11)

$$p(u_{i_t} = y \mid u_\ell = y, u_r \neq y) = \frac{(1 + (1-2\alpha)^{|\ell - i_t|})(1 - (1-2\alpha)^{|r - i_t|})}{2(1 - (1-2\alpha)^{|\ell - r|})}$$

which is greater than $\frac{1}{2}$ if $|\ell - i_t| < |r - i_t|$ and less than $\frac{1}{2}$ if $|\ell - i_t| > |r - i_t|$. Hence, this produces predictions exactly in accordance with the nearest neighbour scheme. We also have more simply that for all $i_t, \ell$ and $r$ and $\alpha < \frac{1}{2}$

$$p(u_{i_t} = y \mid u_\ell = y, u_r = y) > \frac{1}{2}, \quad \text{and} \quad p(u_{i_t} = y \mid u_\ell = y) > \frac{1}{2}.$$

This proves the lemma for all cases. □

A direct application of the Halving algorithm mistake bound (5) now gives

$$M \leq \log_2\left(\frac{1}{p(\boldsymbol{u})}\right) = \log_2\left(\frac{2}{\alpha^{\Phi_\mathcal{P}(\boldsymbol{u})}(1-\alpha)^{n-1-\Phi_\mathcal{P}(\boldsymbol{u})}}\right)$$

where $\boldsymbol{u}$ is any labelling consistent with the trial sequence. We choose $\alpha = \min(\frac{\Phi_{\mathcal{P}}(\boldsymbol{u})}{n-1}, \frac{1}{2})$ (note that the bound is vacuous when $\frac{\Phi_{\mathcal{P}}(\boldsymbol{u})}{n-1} > \frac{1}{2}$ since $M$ is necessarily upper bounded by $n$) giving

$$
\begin{aligned}
M &\leq \Phi_{\mathcal{P}}(\boldsymbol{u}) \log_2\left(\frac{n-1}{\Phi_{\mathcal{P}}(\boldsymbol{u})}\right) + (n-1-\Phi_{\mathcal{P}}(\boldsymbol{u})) \log_2\left(1 + \frac{\Phi_{\mathcal{P}}(\boldsymbol{u})}{n-1-\Phi_{\mathcal{P}}(\boldsymbol{u})}\right) + 1 \\
&\leq \Phi_{\mathcal{P}}(\boldsymbol{u}) \log_2\left(\frac{n-1}{\Phi_{\mathcal{P}}(\boldsymbol{u})}\right) + \frac{\Phi_{\mathcal{P}}(\boldsymbol{u})}{\ln 2} + 1.
\end{aligned}
$$

This proves the theorem. $\qquad\square$

The nearest neighbour algorithm can predict the labelling of any graph $\mathcal{G} = (V, E)$, by first transferring the data representation to that of a spine $\mathcal{S}$ of $\mathcal{G}$, as presented in Section 4. We now apply the above argument to this method and immediately deduce our first main result.

**Theorem 7.** *Given the task of predicting the labelling of any unweighted, connected, $n$-vertex graph $\mathcal{G} = (V, E)$ in the online framework, the number of mistakes, $M$, incurred by the nearest neighbour algorithm operating on a spine $\mathcal{S}$ of $\mathcal{G}$ satisfies*

$$
M \leq 2\Phi_{\mathcal{G}}(\boldsymbol{u}) \max\left[0, \log_2\left(\frac{n-1}{2\Phi_{\mathcal{G}}(\boldsymbol{u})}\right)\right] + \frac{2\Phi_{\mathcal{G}}(\boldsymbol{u})}{\ln 2} + 1, \tag{12}
$$

*where $\boldsymbol{u} \in \{-1, 1\}^n$ is any labelling consistent with the trial sequence.*

*Proof.* Theorem 4 gives bound (4) for predicting on any path, hence $M \leq \Phi_{\mathcal{S}}(\boldsymbol{u}) \log_2\left(\frac{n-1}{\Phi_{\mathcal{S}}(\boldsymbol{u})}\right) + \frac{\Phi_{\mathcal{S}}(\boldsymbol{u})}{\ln 2} + 1$. Since this is an increasing function of $\Phi_{\mathcal{S}}(\boldsymbol{u})$ for $\Phi_{\mathcal{S}}(\boldsymbol{u}) \leq n-1$ and is vacuous at $\Phi_{\mathcal{S}}(\boldsymbol{u}) \geq n-1$ ($M$ is necessarily upper bounded by $n$) we upper bound substituting $\Phi_{\mathcal{S}}(\boldsymbol{u}) \leq 2\Phi_{\mathcal{G}}(\boldsymbol{u})$ (equation (3)). $\qquad\square$

We observe that predicting with the spine is a minimax improvement over Laplacian minimal seminorm interpolation. Recall Theorem 3, there we showed that there exists a trial sequence such that Laplacian minimal semi-norm interpolation incurs $\theta(\sqrt{n})$ mistakes. In fact this trivially generalizes to $\theta(\sqrt{\Phi_{\mathcal{G}}(\boldsymbol{u})n})$ mistakes by creating a colony of $\Phi_{\mathcal{G}}(\boldsymbol{u})$ octopi then identifying each previously separate head vertex as a single central vertex. The upper bound (12) is smaller than the prior lower bound.

The computational complexity for this algorithm is $O(|E| + |V| \ln |V|)$ time. We compute the spine in $O(|E|)$ time by simply listing vertices in the order in which they are first visited during a depth-first search traversal of $\mathcal{G}$. Using online 1-NN requires $O(|V| \ln |V|)$ time to predict an arbitrary vertex sequence using a self-balancing binary search tree (e.g., a red-black tree) as the insertion of each vertex into the tree and determination of the nearest left and right neighbour is $O(\ln |V|)$.

## 6 Prediction with a binary support tree

The Pounce online label prediction algorithm [7] is designed to exploit cluster structure of a graph $\mathcal{G} = (V, E)$ and achieves the following mistake bound

$$
M \leq \mathcal{N}(X, \rho, r_{\mathcal{G}}) + 4\Phi_{\mathcal{G}}(\boldsymbol{u})\rho + 1, \tag{13}
$$

for any $\rho > 0$. Here, $\boldsymbol{u} \in \mathbb{R}^n$ is any labelling consistent with the trial sequence, $X = \{v_{i_1}, v_{i_2}, \dots\} \subseteq V$ is the set of inputs and $\mathcal{N}(X, \rho, r_{\mathcal{G}})$ is a covering number – the minimum number of balls of resistance diameter $\rho$ (see Section 2) required to cover $X$. The mistake bound (13) can be preferable to (12) whenever the inputs are sufficiently clustered and so has a cover of small diameter sets. For example, consider two $(m+1)$-cliques, one labeled "+1", one "−1" with $cm$ arbitrary interconnecting edges ($c \geq 1$) here the bound (12) is vacuous while (13) is $M \leq 8c+3$ (with $\rho = \frac{2}{m}$, $\mathcal{N}(X, \rho, r_{\mathcal{G}}) = 2$, and $\Phi_{\mathcal{G}}(\boldsymbol{u}) = cm$). An input space $V$ may have both local cluster structure yet have a large diameter. Imagine a "universe" such that points are distributed into many dense clusters such that some sets of clusters are tightly packed but overall the distribution is quite diffuse. A given "problem" $X \subseteq V$ may then be centered on a few clusters or alternatively encompass the entire space. Thus, for practical purposes, we would like a prediction algorithm

which achieves the "best of both worlds", that is a mistake bound which is no greater, in order of magnitude, than the maximum of (12) and (13). The rest of this paper is directed toward this goal.

We now introduce the notion of binary support tree, formalise the Pounce method in the support tree setting and then prove the desired result.

**Definition 8.** *Given a graph $\mathcal{G} = (V, E)$, with $|V| = n$, and spine $\mathcal{S}$, we define a binary support tree of $\mathcal{G}$ to be any binary tree $\mathcal{T} = (V_{\mathcal{T}}, E_{\mathcal{T}})$ of least possible depth, $D$, whose leaves are the vertices of $\mathcal{S}$, in order. Note that $D < \log_2(n) + 1$.*

We show that there is a weighting of the support tree which ensures that the resistance diameter of the support tree is small, but also such that any labelling of the leaf vertices can be extended to the support tree such that its cut size remains small. This enables effective learning via the support tree. A related construction has been used to build preconditioners for solving linear systems [6].

**Lemma 9.** *Given any spine graph $\mathcal{S} = (V, E)$ with $|V| = n$, and labelling $\boldsymbol{u} \in \{-1, 1\}^n$, with support tree $\mathcal{T} = (V_{\mathcal{T}}, E_{\mathcal{T}})$, there exists a weighting $\boldsymbol{A}$ of $\mathcal{T}$, and a labelling $\bar{\boldsymbol{u}} \in [-1, 1]^{|V_{\mathcal{T}}|}$ of $\mathcal{T}$ such that $\bar{\boldsymbol{u}}$ and $\boldsymbol{u}$ are identical on $V$, $\Phi_{\mathcal{T}}(\bar{\boldsymbol{u}}) < \Phi_{\mathcal{S}}(\boldsymbol{u})$ and $R_{\mathcal{T}} \leq (\log_2 n + 1)(\log_2 n + 4)(\log_2(\log_2 n + 2))^2$.*

*Proof.* Let $v_r$ be the root vertex of $\mathcal{T}$. Suppose each edge $(i, j) \in E_{\mathcal{T}}$ has a weight $A_{ij}$, which is a function of the edge's depth $d = \max\{d_{\mathcal{T}}(v_i, v_r), d_{\mathcal{T}}(v_j, v_r)\}$, $A_{ij} = W(d)$ where $d_{\mathcal{T}}(v, v')$ is the number of edges in the shortest path from $v$ to $v'$. Consider the unique labelling $\bar{\boldsymbol{u}}$ such that, for $1 \leq i \leq n$ we have $\bar{u}_i = u_i$ and such that for every other vertex $v_p \in V_{\mathcal{T}}$, with child vertices $v_{c_1}, v_{c_2}$, we have $\bar{u}_p = \frac{\bar{u}_{c_1} + \bar{u}_{c_2}}{2}$, or $\bar{u}_p = \bar{u}_c$ in the case where $v_p$ has only one child, $v_c$. Suppose the edges $(p, c_1), (p, c_2) \in E_{\mathcal{T}}$ are at some depth $d$ in $\mathcal{T}$, and let $V' \subset V$ correspond to the leaf vertices of $\mathcal{T}$ descended from $v_p$. Define $\Phi_{\mathcal{S}}(\boldsymbol{u}_{V'})$ to be the cut of $\boldsymbol{u}$ restricted to vertices in $V'$. If $\bar{u}_{c_1} = \bar{u}_{c_2}$ then $(\bar{u}_p - \bar{u}_{c_1})^2 + (\bar{u}_p - \bar{u}_{c_2})^2 = 0 \leq 2\Phi_{\mathcal{S}}(\boldsymbol{u}_{V'})$, and if $\bar{u}_{c_1} \neq \bar{u}_{c_2}$ then $(\bar{u}_p - \bar{u}_{c_1})^2 + (\bar{u}_p - \bar{u}_{c_2})^2 \leq 2 \leq 2\Phi_{\mathcal{S}}(\boldsymbol{u}_{V'})$. Hence

$$W(d)\left((\bar{u}_p - \bar{u}_{c_1})^2 + (\bar{u}_p - \bar{u}_{c_2})^2\right) \leq 2W(d)\Phi_{\mathcal{S}}(\boldsymbol{u}_{V'}) \tag{14}$$

(a similar inequality is trivial in the case that $v_p$ has only one child). Since the sets of leaf descendants of all vertices at depth $d$ form a partition of $V$, summing (14) first over all parent nodes at a given depth and then over all integers $d \in [1, D]$ gives

$$4\Phi_{\mathcal{T}}(\bar{\boldsymbol{u}}) \leq 2\sum_{d=1}^{D} W(d)\Phi_{\mathcal{S}}(\boldsymbol{u}). \tag{15}$$

We then choose

$$W(d) = \frac{1}{(d+1)(\log_2(d+1))^2} \tag{16}$$

and note that $\sum_{d=1}^{\infty} \frac{1}{(d+1)(\log_2(d+1))^2} \leq \frac{1}{2} + \ln^2 2 \int_2^{\infty} \frac{1}{x \ln^2 x} \mathrm{d}x = \frac{1}{2} + \ln 2 < 2$.

Further, $R_{\mathcal{T}} = 2\sum_{d=1}^{D}(d+1)(\log_2(d+1))^2 \leq D(D+3)(\log_2(D+1))^2$ and so $D \leq \log_2 n + 1$ gives the resistance bound. $\square$

**Definition 10.** *Given the task of predicting the labelling of an unweighted graph $\mathcal{G} = (V, E)$ the augmented Pounce algorithm proceeds as follows: An augmented graph $\bar{\mathcal{G}} = (\bar{V}, \bar{E})$ is formed by attaching a binary support tree of $\mathcal{G}$, with weights defined as in (16), to $\mathcal{G}$; formally let $\mathcal{T} = (V_{\mathcal{T}}, E_{\mathcal{T}})$ be such a binary support tree of $\mathcal{G}$, then $\bar{\mathcal{G}} = (V_{\mathcal{T}}, E \cup E_{\mathcal{T}})$. The Pounce algorithm is then used to predict the (partial) labelling defined on $\bar{\mathcal{G}}$.*

**Theorem 11.** *Given the task of predicting the labelling of any unweighted, connected, $n$-vertex graph $\mathcal{G} = (V, E)$ in the online framework, the number of mistakes, $M$, incurred by the augmented Pounce algorithm satisfies*

$$M \leq \min_{\rho > 0}\{\mathcal{N}(X, \rho, r_{\mathcal{G}}) + 12\Phi_{\mathcal{G}}(\boldsymbol{u})\rho\} + 1, \tag{17}$$

*where $\mathcal{N}(X, \rho, r_{\mathcal{G}})$ is the covering number of the input set $X = \{v_{i_1}, v_{i_2}, \dots\} \subseteq V$ relative to the resistance distance $r_{\mathcal{G}}$ of $\mathcal{G}$ and $\boldsymbol{u} \in \mathbb{R}^n$ is any labelling consistent with the trial sequence. Furthermore,*

$$M \leq 12\Phi_{\mathcal{G}}(\boldsymbol{u})(\log_2 n + 1)(\log_2 n + 4)(\log_2(\log_2 n + 2))^2 + 2. \tag{18}$$

*Proof.* Let $\boldsymbol{u}$ be some labelling consistent with the trial sequence. By (3) we have that $\Phi_{\mathcal{S}}(\boldsymbol{u}) \leq 2\Phi_{\mathcal{G}}(\boldsymbol{u})$ for any spine $\mathcal{S}$ of $\mathcal{G}$. Moreover, by the arguments in Lemma 9 there exists some labelling $\bar{\boldsymbol{u}}$ of the weighted support tree $\mathcal{T}$ of $\mathcal{G}$, consistent with $\boldsymbol{u}$ on $V$, such that $\Phi_{\mathcal{T}}(\bar{\boldsymbol{u}}) < \Phi_{\mathcal{S}}(\boldsymbol{u})$. We then have

$$\Phi_{\bar{\mathcal{G}}}(\bar{\boldsymbol{u}}) = \Phi_{\mathcal{T}}(\bar{\boldsymbol{u}}) + \Phi_{\mathcal{G}}(\boldsymbol{u}) < 3\Phi_{\mathcal{G}}(\boldsymbol{u}). \tag{19}$$

By Rayleigh's monotonicity law the addition of the support tree does not increase the resistance between any vertices on $\mathcal{G}$, hence

$$\mathcal{N}(X, \rho, r_{\bar{\mathcal{G}}}) \leq \mathcal{N}(X, \rho, r_{\mathcal{G}}). \tag{20}$$

Combining inequalities (19) and (20) with the pounce bound (13) for predicting $\bar{\boldsymbol{u}}$ on $\bar{\mathcal{G}}$, yields

$$M \leq \mathcal{N}(X, \rho, r_{\bar{\mathcal{G}}}) + 4\Phi_{\bar{\mathcal{G}}}(\bar{\boldsymbol{u}})\rho + 1 \leq \mathcal{N}(X, \rho, r_{\mathcal{G}}) + 12\Phi_{\mathcal{G}}(\boldsymbol{u})\rho + 1.$$

which proves (17). We prove (18) by covering $\bar{\mathcal{G}}$ with single ball so that $M \leq 4\Phi_{\bar{\mathcal{G}}}(\bar{\boldsymbol{u}})R_{\bar{\mathcal{G}}} + 2 \leq 12\Phi_{\mathcal{G}}(\boldsymbol{u})R_{\mathcal{T}} + 2$ and the result follows from the bound on $R_{\mathcal{T}}$ in Lemma 9. □

## 7   Conclusion

We have explored a deficiency with existing online techniques for predicting the labelling of a graph. As a solution, we have presented an approximate cut-preserving embedding of any graph $\mathcal{G} = (V, E)$ into a simple path graph, which we call a spine, such that an implementation of the 1-nearest-neighbours algorithm is an efficient realisation of a Bayes optimal classifier. This therefore achieves a mistake bound which is logarithmic in the size of the vertex set for any graph, and the complexity of our algorithm is of $O(|E| + |V| \ln |V|)$. We further applied the insights gained to a second algorithm – an augmentation of the Pounce algorithm, which achieves a polylogarithmic performance guarantee, but can further take advantage of clustered data, in which case its bound is relative to any cover of the graph.

## References

[1] J. M. Barzdin and R. V. Frievald. On the prediction of general recursive functions. *Soviet Math. Doklady*, 13:1224–1228, 1972.

[2] M. Belkin and P. Niyogi. Semi-supervised learning on riemannian manifolds. *Machine Learning*, 56:209–239, 2004.

[3] A. Blum and S. Chawla. Learning from labeled and unlabeled data using graph mincuts. In *Proc. 18th International Conf. on Machine Learning*, pages 19–26. Morgan Kaufmann, San Francisco, CA, 2001.

[4] P. Doyle and J. Snell. *Random walks and electric networks*. Mathematical Association of America, 1984.

[5] J. Fakcharoenphol and B. Kijsirikul. Low congestion online routing and an improved mistake bound for online prediction of graph labeling. *CoRR*, abs/0809.2075, 2008.

[6] K. Gremban, G. Miller, and M. Zagha. Performance evaluation of a new parallel preconditioner. *Parallel Processing Symposium, International*, 0:65, 1995.

[7] M. Herbster. Exploiting cluster-structure to predict the labeling of a graph. In *The 19th International Conference on Algorithmic Learning Theory*, pages 54–69, 2008.

[8] M. Herbster and M. Pontil. Prediction on a graph with a perceptron. In B. Schölkopf, J. Platt, and T. Hoffman, editors, *Advances in Neural Information Processing Systems 19*, pages 577–584. MIT Press, Cambridge, MA, 2007.

[9] M. Herbster, M. Pontil, and L. Wainer. Online learning over graphs. In *ICML '05: Proceedings of the 22nd international conference on Machine learning*, pages 305–312, New York, NY, USA, 2005. ACM.

[10] R. Kinderman and J. L. Snell. *Markov Random Fields and Their Applications*. Amer. Math. Soc., Providence, RI, 1980.

[11] D. Klein and M. Randić. Resistance distance. *Journal of Mathematical Chemistry*, 12(1):81–95, 1993.

[12] N. Littlestone. Learning when irrelevant attributes abound: A new linear-threshold algorithm. *Machine Learning*, 2:285–318, 1988.

[13] K. Pelckmans and J. A. Suykens. An online algorithm for learning a labeling of a graph. In *In Proceedings of the 6th International Workshop on Mining and Learning with Graphs*, 2008.

[14] X. Zhu, Z. Ghahramani, and J. Lafferty. Semi-supervised learning using gaussian fields and harmonic functions. In *20-th International Conference on Machine Learning (ICML-2003)*, pages 912–919, 2003.
